# Learning Sequential Tasks by Incrementally Adding Higher Orders

**Mark Ring**
Department of Computer Sciences, Taylor 2.124
University of Texas at Austin
Austin, Texas 78712
(ring@cs.utexas.edu)

## Abstract

An incremental, higher-order, non-recurrent network combines two properties found to be useful for learning sequential tasks: higher-order connections and incremental introduction of new units. The network adds higher orders when needed by adding new units that dynamically modify connection weights. Since the new units modify the weights at the next time-step with information from the previous step, temporal tasks can be learned without the use of feedback, thereby greatly simplifying training. Furthermore, a theoretically unlimited number of units can be added to reach into the arbitrarily distant past. Experiments with the Reber grammar have demonstrated speedups of two orders of magnitude over recurrent networks.

## 1   INTRODUCTION

Second-order recurrent networks have proven to be very powerful [8], especially when trained using complete back propagation through time [1, 6, 14]. It has also been demonstrated by Fahlman that a recurrent network that incrementally adds nodes during training—his Recurrent Cascade-Correlation algorithm [5]—can be superior to non-incremental, recurrent networks [2, 4, 11, 12, 15].

The incremental, higher-order network presented here combines advantages of both of these approaches in a non-recurrent network. This network (a simplified, con-

tinuous version of that introduced in [9]), adds higher orders when they are needed by the system to solve its task. This is done by adding new units that dynamically modify connection weights. The new units modify the weights at the next time-step with information from the last, which allows temporal tasks to be learned without the use of feedback.

## 2    GENERAL FORMULATION

Each unit $(U)$ in the network is either an input $(I)$, output $(O)$, or high-level $(L)$ unit.

$$U^i(t) \overset{def}{=} \text{value of } i\text{th unit at time } t.$$

$$I^i(t) \overset{def}{=} U^i(t) \text{ where } i \text{ is an input unit.}$$

$$O^i(t) \overset{def}{=} U^i(t) \text{ where } i \text{ is an output unit.}$$

$$T^i(t) \overset{def}{=} \text{Target value for } O^i(t) \text{ at time } t.$$

$$L^i_{xy}(t) \overset{def}{=} U^i(t) \text{ where } i \text{ is the higher-order unit that}$$
$$\text{modifies weight } w_{xy} \text{ at time } t.^1$$

The output and high-level units are collectively referred to as non-input $(N)$ units:

$$N^i(t) \overset{def}{=} \begin{cases} O^i(t) & \text{if } U^i \equiv O^i. \\ L^i_{xy}(t) & \text{if } U^i \equiv L^i_{xy}. \end{cases}$$

In a given time-step, the output and high-level units receive a summed input from the input units.

$$N^i(t) = \sum_j I^j(t)g(i,j,t). \tag{1}$$

$g$ is a gating function representing the weight of a particular connection at a particular time-step. If there is a higher-order unit assigned to that connection, then the input value of that unit is added to the connection's weight at that time-step.[2]

$$g(i,j,t) = \begin{cases} w_{ij}(t) + L^n_{ij}(t-1) & \text{If } L^n_{ij} \text{ exists} \\ w_{ij}(t) & \text{Otherwise} \end{cases} \tag{2}$$

At each time-step, the values of the output units are calculated from the input units and the weights (possibly modified by the activations of the high-level units from the previous time-step). The values of the high-level units are calculated at the same time in the same way. The output units generate the output of the network. The high-level units simply alter the weights at the next time-step. All unit activations can be computed simultaneously since the activations of the $L$ units are not required

until the following time-step. The network is arranged hierarchically in that every higher-order units is always higher in the hierarchy than the units on either side of the weight it affects. Since higher-order units have no outgoing connections, the network is not recurrent. It is therefore impossible for a high-level unit to affect, directly or indirectly, its own input.

There are no hidden units in the traditional sense, and all units have a linear activation function. (This does not imply that non-linear functions cannot be represented, since non-linearities do result from the multiplication of higher-level and input units in equations 1 and 2.)

Learning is done through gradient descent to reduce the sum-squared error.

$$E(t) = \frac{1}{2}\sum_i (T^i(t) - O^i(t))^2$$

$$w_{ij}(t+1) = w_{ij}(t) - \eta \Delta w_{ij}(t)$$

$$\Delta w_{ij}(t) = \sum_{\tau=0}^{t} \frac{\partial E(t)}{\partial w_{ij}(\tau)} , \tag{3}$$

where $\eta$ is the learning rate. Since it may take several time-steps for the value of a weight to affect the network's output and therefore the error, equation 3 can be rewritten as:

$$\Delta w_{ij}(t) = \frac{\partial E(t)}{\partial w_{ij}(t - \tau^i)} , \tag{4}$$

where

$$\tau^i = \begin{cases} 0 & \text{if } U^i \equiv O^i \\ 1 + \tau^x & \text{if } U^i \equiv L_{xy}^i \end{cases}$$

The value $\tau^i$ is constant for any given unit $i$ and specifies how "high" in the hierarchy unit $i$ is. It therefore also specifies how many time-steps it takes for a change in unit $i$'s activation to affect the network's output.

Due to space limitations, the derivation of the gradient is not shown, but is given elsewhere [10]. The resulting weight change rule, however, is:

$$\Delta w_{ij}(t) = I^j(t - \tau^i) \begin{cases} T^i(t) - O^i(t) & \text{If } U^i \equiv O^i \\ \Delta w_{xy}(t) & \text{If } U^i \equiv L_{xy}^i \end{cases} \tag{5}$$

The weights are changed after error values for the output units have been collected. Since each high-level unit is higher in the hierarchy than the units on either side of the weight it affects, weight changes are made bottom up, and the $\Delta w_{xy}(t)$ in equation 5 will already have been calculated at the time $\Delta w_{ij}(t)$ is computed.

The intuition behind the learning rule is that each high-level unit learns to utilize the context from the previous time-step for adjusting the connection it influences at the next time-step so that it can minimize the connection's error in that context. Therefore, if the information necessary to decide the correct value of a connection at one time-step is available at the previous time-step, then that information is used by the higher-order unit assigned to that connection. If the needed information is not available at the previous time-step, then new units may be built to look for the information at still earlier steps. This method concentrating on unexpected events is similar to the "hierarchy of decisions" of Dawkins [3], and the "history compression" of Schmidhuber [13].

## 3    WHEN TO ADD NEW UNITS

A unit is added whenever a weight is being pulled strongly in opposite directions (i.e. when learning is forcing the weight to increase and to decrease at the same time). The unit is created to determine the contexts in which the weight is pulled in each direction. This is done in the following way: Two long-term averages are kept for each connection. The first of these records the average change made to the weight,

$$\overline{\Delta w_{ij}(t)} = \sigma \Delta w_{ij}(t) + (1 - \sigma)\overline{\Delta w_{ij}(t-1)}; \quad 0 \le \sigma \le 1.$$

The second is the long-term mean absolute deviation, given by:

$$\overline{|\Delta w_{ij}(t)|} = \sigma |\Delta w_{ij}(t)| + (1 - \sigma)\overline{|\Delta w_{ij}(t-1)|}; \quad 0 \le \sigma \le 1.$$

The parameter, $\sigma$, specifies the duration of the long-term average. A lower value of $\sigma$ means that the average is kept for a longer period of time. When $\overline{\Delta w_{ij}(t)}$ is small, but $\overline{|\Delta w_{ij}(t)|}$ is large, then the weight is being pulled strongly in conflicting directions, and a new unit is built.

$$\text{if} \qquad \frac{\overline{|\Delta w_{ij}(t)|}}{\epsilon + |\overline{\Delta w_{ij}(t)}|} > \Theta \qquad \text{then build unit } L_{ij}^{N+1},$$

where $\epsilon$ is a small constant that keeps the denominator from being zero, $\Theta$ is a threshold value, and $N$ is the number of units in the network. A related method for adding new units in feed-forward networks was introduced by Wynne-Jones [16].

When a new unit is added, its incoming weights are initially zero. It has no output weights but simply learns to anticipate and reduce the error at each time-step of the weight it modifies. In order to keep the number of new units low, whenever a unit, $L_{ij}^n$ is created, the statistics for all connections into the destination unit ($U^i$) are reset: $\overline{|\Delta w_{ij}(t)|} \leftarrow 0.0$ and $\overline{\Delta w_{ij}(t)} \leftarrow 1.0$.

## 4    RESULTS

The Reber grammar is a small finite-state grammar of the following form:

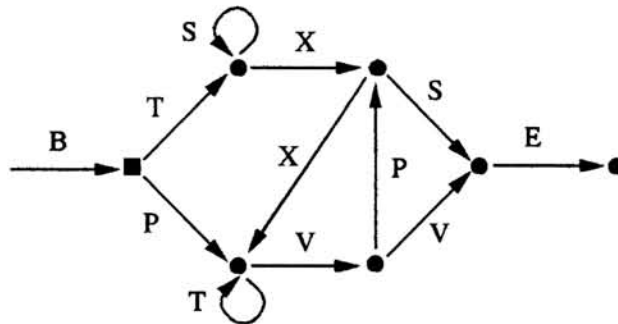

Transitions from one node to the next are made by way of the labeled arcs. The task of the network is: given as input the label of the arc just traversed, predict

|  |  | Elman Network | RTRL | Recurrent Cascade Correlation | Incremental Higher-Order Network |
|---|---|---|---|---|---|
| Sequences Seen: | Mean |  |  | 25,000 | 206 |
|  | Best | 20,000 | 19,000 |  | 176 |
| "Hidden" Units |  | 15 | 2 | 2-3 | 40 |

Table 1: The incremental higher-order network is compared against recurrent networks on the Reber grammar. The results for the recurrent networks are quoted from other sources [2, 5]. The mean and/or best performance is shown when available. RTRL is the real-time recurrent learning algorithm [15].

the arc that will be traversed next. A training sequence, or *string*, is generated by starting with a B transition and then randomly choosing an arc leading away from the current state until the final state is reached. Both inputs and outputs are encoded locally, so that there are seven output units (one each for B, T, S, X, V, P, and E) and eight input units (the same seven plus one bias unit). The network is considered correct if its highest activated outputs correspond to the arcs that can be traversed from the current state. Note that the current state cannot be determined from the current input alone.

An Elman-type recurrent network was able to learn this task after 20,000 string presentations using 15 hidden units [2]. (The correctness criteria for the Elman net was slightly more stringent than that described in the previous paragraph.) Recurrent Cascade-Correlation (RCC) was able to learn this task using only two or three hidden units in an average of 25,000 string presentations [5].

The incremental, higher-order network was trained on a continuous stream of input: the network was not reset before beginning a new string. Training was considered to be complete only after the network had correctly classified 100 strings in a row. Using this criterion, the network completed training after an average of 206.3 string presentations with a standard deviation of 16.7. It achieved perfect generalization on test sets of 128 randomly generated strings in all ten runs. Because the Reber grammar is stochastic, a ceiling of 40 higher-order units was imposed on the network to prevent it from continually creating new units in an attempt to outguess the random number generator.

Complete results for the network on the Reber grammar task are given in table 1. The parameter settings were: $\eta = 0.04, \sigma = 0.08, \Theta = 1.0, \epsilon = 0.1$ and Bias = 0.0. (The network seemed to perform better with no bias unit.)

The network has also been tested on the "variable gap" tasks introduced by Mozer [7], as shown in figure 1. These tasks were intended to test performance of networks over long time-delays. Two sequences are alternately presented to the network. Each sequence begins with an X or a Y and is followed by a fixed string of characters with an X or a Y inserted some number of time-steps from the beginning. In figure 1 the number of time-steps, or "gap", is 2. The only difference between the two sequences is that the first begins with an X and repeats the X after the gap, while the second begins with a Y and repeats the Y after the gap. The network must learn to predict the next item in the sequence given the current item as input

| Time-step: | 0 | 1 | 2 | 3 | 4 | 5 | 6 | 7 | 8 | 9 | 10 | 11 | 12 |
|---|---|---|---|---|---|---|---|---|---|---|---|---|---|
| Sequence 1: | X | a | b | X | c | d | e | f | g | h | i | j | k |
| Sequence 2: | Y | a | b | Y | c | d | e | f | g | h | i | j | k |

Figure 1: An example of a "variable gap" training sequence [7]. One item is presented to the network at each time-step. The target is the next item in the sequence. Here the "gap" is two, because there are two items in the sequence between the first X or Y and the second X or Y. In order to correctly predict the second X or Y, the network must remember how the sequence began.

(where all inputs are locally encoded). In order for the network to predict the second occurrence of the X or Y, it must remember how the sequence began. The length of the gap can be increased in order to create tasks of greater difficulty.

Results of the "gap" tasks are given in table 2. The values for the standard recurrent network and for Mozer's own variation are quoted from Mozer's paper [7]. The incremental higher-order net had no difficulty with gaps up to 24, which was the largest gap I tested. The same string was used for all tasks (except for the position of the second X or Y), and had no repeated characters (again with the exception of the X and Y). The network continued to scale linearly with every gap size both in terms of units and epochs required for training. Because these tasks are not stochastic, the network always stopped building units as soon as it had created those necessary to solve each task.

The parameter settings were: $\eta = 1.5, \sigma = 0.2, \Theta = 1.0, \epsilon = 0.1$ and Bias $= 0.0$. The network was considered to have correctly predicted an element in the sequence if the most strongly activated output unit was the unit representing the correct prediction. The sequence was considered correctly predicted if all elements (other than the initial X or Y) were correctly predicted.

| Gap | Mean number of Training sets required by: | | | |
|---|---|---|---|---|
| | Standard Recurrent Net | Mozer Network | Incremental Higher-Order Net | Units Created |
| 2 | 468 | 328 | 4 | 10 |
| 4 | 7406 | 584 | 6 | 15 |
| 6 | 9830 | 992 | 8 | 19 |
| 8 | > 10000 | 1312 | 10 | 23 |
| 10 | > 10000 | 1630 | 12 | 27 |
| 24 | | | 26 | 49 |

Table 2: A comparison on the "gap" tasks of a standard recurrent-network and a network devised specifically for long time-delays (quoted from Mozer [7], who reported results for gaps up to ten) against an incremental higher-order network. The last column is the number of units created by the incremental higher-order net.

## 5 CONCLUSIONS

The incremental higher-order network performed much better than the networks that it was compared against on these tiny tests. A few caveats are in order, however. First, the parameters given for the tasks above were customized for those tasks. Second, the network may add a large number of new units if it contains many context-dependent events or if it is inherently stochastic. Third, though the network in principle can build an ever larger hierarchy that searches further and further back in time for a context that will predict what a connection's weight should be, many units may be needed to bridge a long time-gap. Finally, once a bridge across a time-delay is created, it does not generalize to other time-delays.

On the other hand, the network learns very fast due to its simple structure that adds high-level units only when needed. Since there is no feedback (i.e. no unit ever produces a signal that will ever feed back to itself), learning can be done without back propagation through time. Also, since the outputs and high-level units have a fan-in equal to the number of inputs only, the number of connections in the system is much smaller than the number of connections in a traditional network with the same number of hidden units.

Finally, the network can be thought of as a system of continuous-valued condition-action rules that are inserted or removed depending on another set of such rules that are in turn inserted or removed depending on another set, etc. When new rules (new units) are added, they are initially invisible to the system, (i.e., they have no effect), but only gradually learn to have an effect as the opportunity to decrease error presents itself.

### Acknowledgements

This work was supported by NASA Johnson Space Center Graduate Student Researchers Program training grant, NGT 50594. I would like to thank Eric Hartman, Kadir Liano, and my advisor Robert Simmons for useful discussions and helpful comments on drafts of this paper. I would also like to thank Pavilion Technologies, Inc. for their generous contribution of computer time and office space required to complete much of this work.

## Footnotes

[1] A connection may be modified by at most one $L$ unit. Therefore $L^i$, $L_{xy}$, and $L^i_{xy}$ are identical but used as appropriate for notational convenience.

[2] It can be seen that this is a higher-order connection in the usual sense if one substitutes the right-hand side of equation 1 for $L^n_{ij}$ in equation 2 and then replaces $g$ in equation 1 with the result. In fact, as the network increases in height, ever higher orders are introduced, while lower orders are preserved.

### References

[1] Jonathan Richard Bachrach. *Connectionist Modeling and Control of Finite State Environments*. PhD thesis, Department of Computer and Information Sciences, University of Massachusetts, February 1992.

[2] Axel Cleeremans, David Servan-Schreiber, and James L. McClelland. Finite state automata and simple recurrent networks. *Neural Computation*, 1(3):372–381, 1989.

[3] Richard Dawkins. Hierarchical organisation: a candidate principle for ethology. In P. P. G. Bateson and R. A. Hinde, editors, *Growing Points in Ethology*, pages 7–54, Cambridge, 1976. Cambridge University Press.

[4] Jeffrey L. Elman. Finding structure in time. CRL Technical Report 8801, University of California, San Diego, Center for Research in Language, April 1988.

[5] Scott E. Fahlman. The recurrent cascade-correlation architecture. In R. P. Lippmann, J. E. Moody, and D. S. Touretzky, editors, *Advances in Neural Information Processing Systems 3*, pages 190–196, San Mateo, California, 1991. Morgan Kaufmann Publishers.

[6] C. L. Giles, C. B. Miller, D. Chen, G. Z. Sun, H. H. Chen, and Y. C. Lee. Extracting and learning an unknown grammar with recurrent neural networks. In J. E. Moody, S. J. Hanson, and R. P. Lippman, editors, *Advances in Neural Information Processing Systems 4*, pages 317–324, San Mateo, California, 1992. Morgan Kaufmann Publishers.

[7] Michael C. Mozer. Induction of multiscale temporal structure. In John E. Moody, Steven J. Hanson, and Richard P. Lippmann, editors, *Advances in Neural Information Processing Systems 4*, pages 275–282, San Mateo, California, 1992. Morgan Kaufmann Publishers.

[8] Jordan B. Pollack. The induction of dynamical recognizers. *Machine Learning*, 7:227–252, 1991.

[9] Mark B. Ring. Incremental development of complex behaviors through automatic construction of sensory-motor hierarchies. In Lawrence A. Birnbaum and Gregg C. Collins, editors, *Machine Learning: Proceedings of the Eighth International Workshop (ML91)*, pages 343–347. Morgan Kaufmann Publishers, June 1991.

[10] Mark B. Ring. Sequence learning with incremental higher-order neural networks. Technical Report AI 93–193, Artificial Intelligence Laboratory, University of Texas at Austin, January 1993.

[11] A. J. Robinson and F. Fallside. The utility driven dynamic error propagation network. Technical Report CUED/F-INFENG/TR.1, Cambridge University Engineering Department, 1987.

[12] D. E. Rumelhart, G. E. Hinton, and R. J. Williams. Learning internal representations by error propagation. In D. E. Rumelhart and J. L. McClelland, editors, *Parallel Distributed Processing: Explorations in the Microstructure of Cognition. V1: Foundations*. MIT Press, 1986.

[13] Jürgen Schmidhuber. Learning unambiguous reduced sequence descriptions. In J. E. Moody, S. J. Hanson, and R. P. Lippman, editors, *Advances in Neural Information Processing Systems 4*, pages 291–298, San Mateo, California, 1992. Morgan Kaufmann Publishers.

[14] Raymond L. Watrous and Gary M. Kuhn. Induction of finite-state languages using second-order recurrent networks. In J. E. Moody, S. J. Hanson, and R. P. Lippman, editors, *Advances in Neural Information Processing Systems 4*, pages 309–316, San Mateo, California, 1992. Morgan Kaufmann Publishers.

[15] Ronald J. Williams and David Zipser. A learning algorithm for continually running fully recurrent neural networks. *Neural Computation*, 1(2):270–280, 1989.

[16] Mike Wynn-Jones. Node splitting: A constructive algorithm for feed-forward neural networks. *Neural Computing and Applications*, 1(1):17–22, 1993.
